# Distance Metric Learning, with Application to Clustering with Side-Information

**Eric P. Xing, Andrew Y. Ng, Michael I. Jordan and Stuart Russell**
University of California, Berkeley
Berkeley, CA 94720
{epxing,ang,jordan,russell}@cs.berkeley.edu

## Abstract

Many algorithms rely critically on being given a good metric over their inputs. For instance, data can often be clustered in many "plausible" ways, and if a clustering algorithm such as K-means initially fails to find one that is meaningful to a user, the only recourse may be for the user to manually tweak the metric until sufficiently good clusters are found. For these and other applications requiring good metrics, it is desirable that we provide a more systematic way for users to indicate what they consider "similar." For instance, we may ask them to provide examples. In this paper, we present an algorithm that, given examples of similar (and, if desired, dissimilar) pairs of points in $\mathbb{R}^n$, learns a distance metric over $\mathbb{R}^n$ that respects these relationships. Our method is based on posing metric learning as a convex optimization problem, which allows us to give efficient, local-optima-free algorithms. We also demonstrate empirically that the learned metrics can be used to significantly improve clustering performance.

## 1 Introduction

The performance of many learning and datamining algorithms depend critically on their being given a good metric over the input space. For instance, K-means, nearest-neighbors classifiers and kernel algorithms such as SVMs all need to be given good metrics that reflect reasonably well the important relationships between the data. This problem is particularly acute in unsupervised settings such as clustering, and is related to the perennial problem of there often being no "right" answer for clustering: If three algorithms are used to cluster a set of documents, and one clusters according to the authorship, another clusters according to topic, and a third clusters according to writing style, who is to say which is the "right" answer? Worse, if an algorithm were to have clustered by topic, and if we instead wanted it to cluster by writing style, there are relatively few systematic mechanisms for us to convey this to a clustering algorithm, and we are often left tweaking distance metrics by hand.

In this paper, we are interested in the following problem: Suppose a user indicates that certain points in an input space (say, $\mathbb{R}^n$) are considered by them to be "similar." Can we automatically learn a distance metric over $\mathbb{R}^n$ that respects these relationships, i.e., one that assigns small distances between the similar pairs? For instance, in the documents example, we might hope that, by giving it pairs of documents judged to be written in similar styles, it would learn to recognize the critical features for determining style.

One important family of algorithms that (implicitly) learn metrics are the unsupervised ones that take an input dataset, and find an embedding of it in some space. This includes algorithms such as Multidimensional Scaling (MDS) [2], and Locally Linear Embedding (LLE) [9]. One feature distinguishing our work from these is that we will learn a full metric $d : \mathbb{R}^n \times \mathbb{R}^n :\mapsto \mathbb{R}$ over the input space, rather than focusing only on (finding an embedding for) the points in the training set. Our learned metric thus generalizes more easily to previously unseen data. More importantly, methods such as LLE and MDS also suffer from the "no right answer" problem: For example, if MDS finds an embedding that fails to capture the structure important to a user, it is unclear what systematic corrective actions would be available. (Similar comments also apply to Principal Components Analysis (PCA) [7].) As in our motivating clustering example, the methods we propose can also be used in a pre-processing step to help any of these unsupervised algorithms to find better solutions.

In the supervised learning setting, for instance nearest neighbor classification, numerous attempts have been made to define or learn either local or global metrics for classification. In these problems, a clear-cut, supervised criterion—classification error—is available and can be optimized for. (See also [11], for a different way of supervising clustering.) This literature is too wide to survey here, but some relevant examples include [10, 5, 3, 6], and [1] also gives a good overview of some of this work. While these methods often learn good metrics *for classification*, it is less clear whether they can be used to learn good, general metrics for *other* algorithms such as K-means, particularly if the information available is less structured than the traditional, homogeneous training sets expected by them.

In the context of clustering, a promising approach was recently proposed by Wagstaff et al. [12] for clustering with similarity information. If told that certain pairs are "similar" or "dissimilar," they search for a clustering that puts the similar pairs into the same, and dissimilar pairs into different, clusters. This gives a way of using similarity side-information to find clusters that reflect a user's notion of meaningful clusters. But similar to MDS and LLE, the ("instance-level") constraints that they use do not generalize to previously unseen data whose similarity/dissimilarity to the training set is not known. We will later discuss this work in more detail, and also examine the effects of using the methods we propose in conjunction with these methods.

## 2   Learning Distance Metrics

Suppose we have some set of points $\{x_i\}_{i=1}^m \subseteq \mathbb{R}^n$, and are given information that certain pairs of them are "similar":

$$S : \quad (x_i, x_j) \in \mathcal{S} \quad \text{if } x_i \text{ and } x_j \text{ are similar} \tag{1}$$

How can we learn a distance metric $d(x, y)$ between points $x$ and $y$ that respects this; specifically, so that "similar" points end up close to each other?

Consider learning a distance metric of the form

$$d(x, y) = d_A(x, y) = ||x - y||_A = \sqrt{(x - y)^T A (x - y)}. \tag{2}$$

To ensure that this be a metric—satisfying non-negativity and the triangle inequality— we require that $A$ be positive semi-definite, $A \succeq 0$.[1] Setting $A = I$ gives Euclidean distance; if we restrict $A$ to be diagonal, this corresponds to learning a metric in which the different axes are given different "weights"; more generally, $A$ parameterizes a family of Mahalanobis distances over $\mathbb{R}^n$.[2] Learning such a distance metric is also equivalent to finding a rescaling of a data that replaces each point $x$ with $A^{1/2}x$ and applying the

standard Euclidean metric to the rescaled data; this will later be useful in visualizing the learned metrics.

A simple way of defining a criterion for the desired metric is to demand that pairs of points $(x_i, x_j)$ in $\mathcal{S}$ have, say, small squared distance between them: $\text{minimize}_A \sum_{(x_i, x_j) \in \mathcal{S}} ||x_i - x_j||_A^2$. This is trivially solved with $A = 0$, which is not useful, and we add the constraint $\sum_{(x_i, x_j) \in \mathcal{D}} ||x_i - x_j||_A \geq 1$ to ensure that $A$ does not collapse the dataset into a single point. Here, $\mathcal{D}$ can be a set of pairs of points known to be "dissimilar" if such information is explicitly available; otherwise, we may take it to be all pairs not in $\mathcal{S}$. This gives the optimization problem:

$$\min_A \quad \sum_{(x_i, x_j) \in \mathcal{S}} ||x_i - x_j||_A^2 \tag{3}$$

$$\text{s.t.} \quad \sum_{(x_i, x_j) \in \mathcal{D}} ||x_i - x_j||_A \geq 1, \tag{4}$$

$$A \succeq 0. \tag{5}$$

The choice of the constant 1 in the right hand side of (4) is arbitrary but not important, and changing it to any other positive constant $c$ results only in $A$ being replaced by $c^2 A$. Also, this problem has an objective that is linear in the parameters $A$, and both of the constraints are also easily verified to be convex. Thus, the optimization problem is *convex*, which enables us to derive efficient, local-minima-free algorithms to solve it.

We also note that, while one might consider various alternatives to (4), "$\sum_{(x_i, x_j) \in \mathcal{D}} ||x_i - x_j||_A^2 \geq 1$" would not be a good choice despite its giving a simple linear constraint. It would result in $A$ always being rank 1 (i.e., the data are always projected onto a line).[3]

## 2.1 The case of diagonal $A$

In the case that we want to learn a diagonal $A = \text{diag}(A_{11}, A_{22}, \ldots, A_{nn})$, we can derive an efficient algorithm using the Newton-Raphson method. Define

$$g(A) = g(A_{11}, \ldots, A_{nn}) = \sum_{(x_i, x_j) \in \mathcal{S}} ||x_i - x_j||_A^2 - \log \left( \sum_{(x_i, x_j) \in \mathcal{D}} ||x_i - x_j||_A \right)$$

It is straightforward to show that minimizing $g$ (subject to $A \succeq 0$) is equivalent, up to a multiplication of $A$ by a positive constant, to solving the original problem (3–5). We can thus use Newton-Raphson to efficiently optimize $g$.[4]

## 2.2 The case of full $A$

In the case of learning a full matrix $A$, the constraint that $A \succeq 0$ becomes slightly trickier to enforce, and Newton's method often becomes prohibitively expensive (requiring $O(n^6)$ time to invert the Hessian over $n^2$ parameters). Using gradient descent and the idea of iterative projections (e.g., [8]) we derive a different algorithm for this setting.

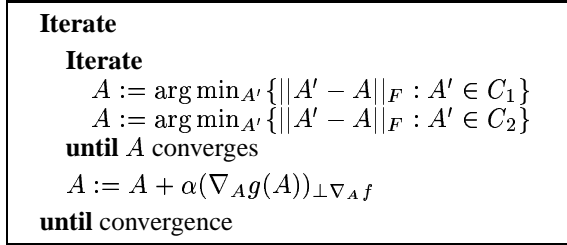

**Iterate**
    **Iterate**
        $A := \arg\min_{A'}\{||A' - A||_F : A' \in C_1\}$
        $A := \arg\min_{A'}\{||A' - A||_F : A' \in C_2\}$
    **until** $A$ converges
    $A := A + \alpha(\nabla_A g(A))_{\perp \nabla_A f}$
**until** convergence

Figure 1: Gradient ascent + Iterative projection algorithm. Here, $||\cdot||_F$ is the Frobenius norm on matrices ($||M||_F = (\sum_i \sum_j M_{ij}^2)^{1/2}$).

We pose the equivalent problem:

$$\max_A \quad g(A) = \sum_{(x_i,x_j)\in\mathcal{D}} ||x_i, x_j||_A \tag{6}$$

$$\text{s.t.} \quad f(A) = \sum_{(x_i,x_j)\in\mathcal{S}} ||x_i, x_j||_A^2 \leq 1 \tag{7}$$

$$A \succeq 0. \tag{8}$$

We will use a gradient ascent step on $g(A)$ to optimize (6), followed by the method of iterative projections to ensure that the constraints (7) and (8) hold. Specifically, we will repeatedly take a gradient step $A := A + \alpha\nabla_A g(A)$, and then repeatedly project $A$ into the sets $C_1 = \{A : \sum_{(x_i,x_j)\in\mathcal{S}} ||x_i - x_j||_A^2 \leq 1\}$ and $C_2 = \{A : A \succeq 0\}$. This gives the algorithm shown in Figure 1.[5]

The motivation for the specific choice of the problem formulation (6–8) is that projecting $A$ onto $C_1$ or $C_2$ can be done inexpensively. Specifically, the first projection step $A := \arg\min_{A'}\{||A' - A||_F^2 : A' \in C_1\}$ involves minimizing a quadratic objective subject to a single linear constraint; the solution to this is easily found by solving (in $O(n^2)$ time) a sparse system of linear equations. The second projection step onto $C_2$, the space of all positive-semi definite matrices, is done by first finding the diagonalization $A = X^T\Lambda X$, where $\Lambda = \text{diag}(\lambda_1, \ldots, \lambda_n)$ is a diagonal matrix of $A$'s eigenvalues and the columns of $X \in \mathbb{R}^{n\times n}$ contains $A$'s corresponding eigenvectors, and taking $A' = X^T\Lambda'X$, where $\Lambda' = \text{diag}(\max\{0, \lambda_1\}, \ldots, \{0, \lambda_n\})$. (E.g., see [4].)

## 3 Experiments and Examples

We begin by giving some examples of distance metrics learned on artificial data, and then show how our methods can be used to improve clustering performance.

### 3.1 Examples of learned distance metrics

Consider the data shown in Figure 2(a), which is divided into two classes (shown by the different symbols and, where available, colors). Suppose that points in each class are "similar" to each other, and we are given $\mathcal{S}$ reflecting this.[6] Depending on whether we learn a diagonal or a full $A$, we obtain:

$$A_{\text{diagonal}} = \begin{bmatrix} 1.036 & 0 & 0 \\ 0 & 1.007 & 0 \\ 0 & 0 & 0 \end{bmatrix}; \quad A_{\text{full}} = \begin{bmatrix} 3.245 & 3.286 & 0.081 \\ 3.286 & 3.327 & 0.082 \\ 0.081 & 0.082 & 0.002 \end{bmatrix}$$

To visualize this, we can use the fact discussed earlier that learning $||\cdot||_A$ is equivalent to finding a rescaling of the data $x \rightarrow A^{1/2}x$, that hopefully "moves" the similar pairs

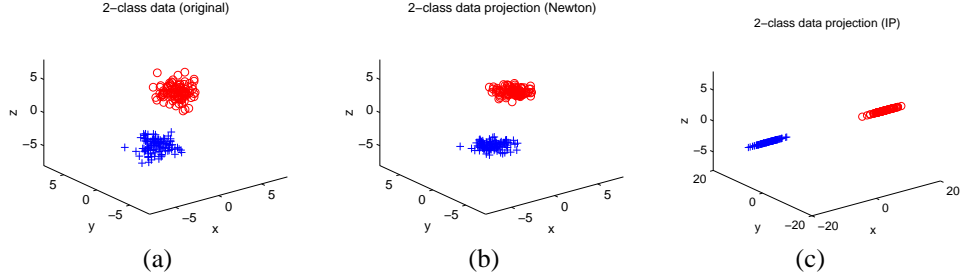

Figure 2: (a) Original data, with the different classes indicated by the different symbols (and colors, where available). (b) Rescaling of data corresponding to learned diagonal $A$. (c) Rescaling corresponding to full $A$.

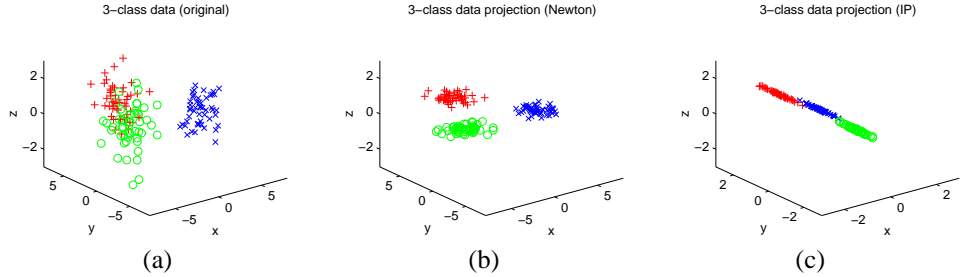

Figure 3: (a) Original data. (b) Rescaling corresponding to learned diagonal $A$. (c) Rescaling corresponding to full $A$.

together. Figure 2(b,c) shows the result of plotting $A^{1/2}x$. As we see, the algorithm has successfully brought together the similar points, while keeping dissimilar ones apart.

Figure 3 shows a similar result for a case of three clusters whose centroids differ only in the x and y directions. As we see in Figure 3(b), the learned diagonal metric correctly ignores the z direction. Interestingly, in the case of a full $A$, the algorithm finds a surprising projection of the data onto a line that still maintains the separation of the clusters well.

### 3.2  Application to clustering

One application of our methods is "clustering with side information," in which we learn a distance metric using similarity information, and cluster data using that metric. Specifically, suppose we are given $\mathcal{S}$, and told that each pair $(x_i, x_j) \in \mathcal{S}$ means $x_i$ and $x_j$ belong to the same cluster. We will consider four algorithms for clustering:

1. K-means using the default Euclidean metric $||x_i - \mu_k||_2^2$ between points $x_i$ and cluster centroids $\mu_k$ to define distortion (and ignoring $\mathcal{S}$).

2. Constrained K-means: K-means but subject to points $(x_i, x_j) \in \mathcal{S}$ always being assigned to the same cluster [12].[7]

3. K-means + metric: K-means but with distortion defined using the distance metric $||x_i - \mu_k||_A^2$ learned from $\mathcal{S}$.

4. Constrained K-means + metric: Constrained K-means using the distance metric learned from $\mathcal{S}$.

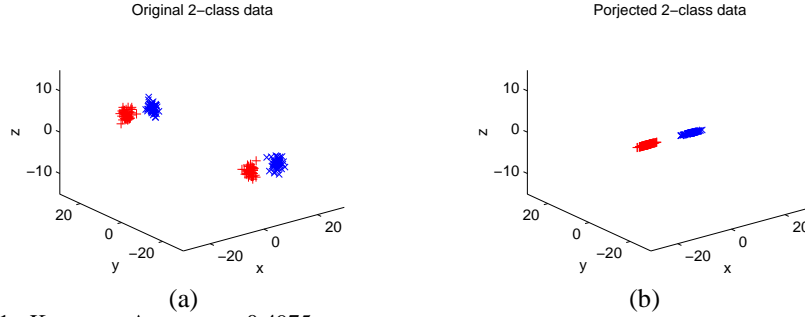

(a)                                      (b)

1. K-means: Accuracy = 0.4975
2. Constrained K-means: Accuracy = 0.5060
3. K-means + metric: Accuracy = 1
4. Constrained K-means + metric: Accuracy = 1

Figure 4: (a) Original dataset (b) Data scaled according to learned metric. ($A_{\text{diagonal}}$'s result is shown, but $A_{\text{full}}$ gave visually indistinguishable results.)

Let $\hat{c}_i$ ($i = 1, \ldots, m$) be the cluster to which point $x_i$ is assigned by an automatic clustering algorithm, and let $c_i$ be some "correct" or desired clustering of the data. Following [**?**], in the case of 2-cluster data, we will measure how well the $\hat{c}_i$'s match the $c_i$'s according to

$$\text{Accuracy} = \sum_{i>j} \frac{1\{1\{c_i = c_j\} = 1\{\hat{c}_i = \hat{c}_j\}\}}{0.5m(m-1)},$$

where $1\{\cdot\}$ is the indicator function ($1\{\text{True}\} = 1$, $1\{\text{False}\} = 0$). This is equivalent to the probability that for two points $x_i$, $x_j$ drawn randomly from the dataset, our clustering $\hat{c}$ agrees with the "true" clustering $c$ on whether $x_i$ and $x_j$ belong to same or different clusters.[8]

As a simple example, consider Figure 4, which shows a clustering problem in which the "true clusters" (indicated by the different symbols/colors in the plot) are distinguished by their $x$-coordinate, but where the data in its original space seems to cluster much better according to their $y$-coordinate. As shown by the accuracy scores given in the figure, both K-means and constrained K-means failed to find good clusterings. But by first learning a distance metric and then clustering according to that metric, we easily find the correct clustering separating the true clusters from each other. Figure 5 gives another example showing similar results.

We also applied our methods to 9 datasets from the UC Irvine repository. Here, the "true clustering" is given by the data's class labels. In each, we ran one experiment using "little" side-information $\mathcal{S}$, and one with "much" side-information. The results are given in Figure 6.[9]

We see that, in almost every problem, using a learned diagonal or full metric leads to significantly improved performance over naive K-means. In most of the problems, using a learned metric with constrained K-means (the 5th bar for diagonal $A$, 6th bar for full $A$) also outperforms using constrained K-means alone (4th bar), sometimes by a very large

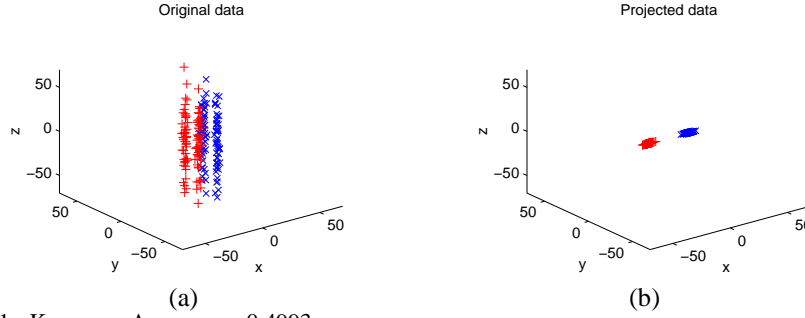

Original data        Projected data

(a)                  (b)

1. K-means: Accuracy = 0.4993
2. Constrained K-means: Accuracy = 0.5701
3. K-means + metric: Accuracy = 1
4. Constrained K-means + metric: Accuracy = 1

Figure 5: (a) Original dataset (b) Data scaled according to learned metric. ($A_\text{diagonal}$'s result is shown, but $A_\text{full}$ gave visually indistinguishable results.)

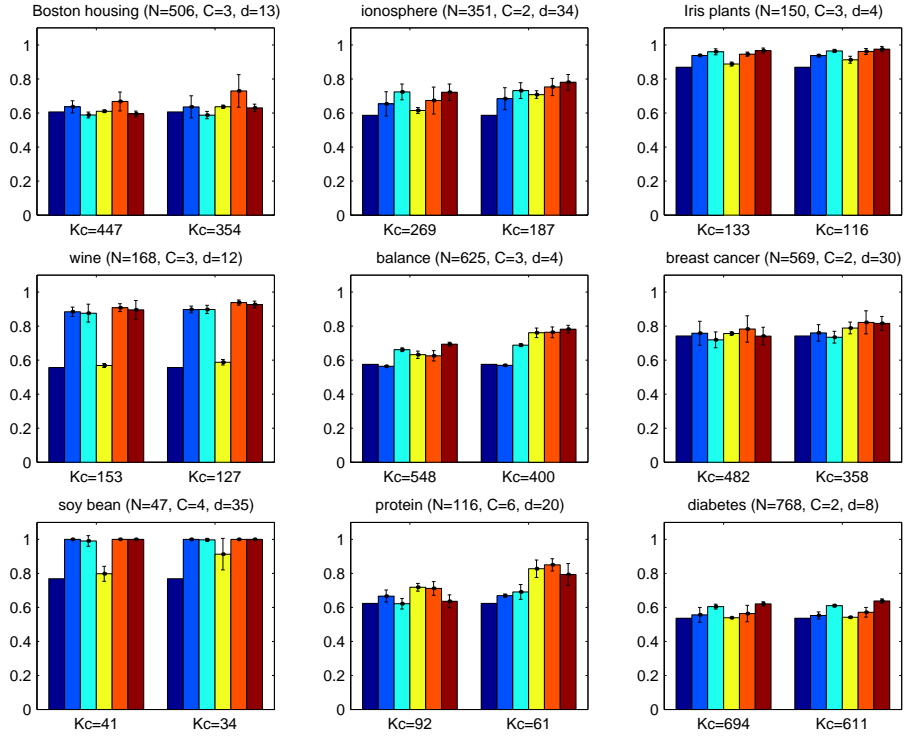

Figure 6: Clustering accuracy on 9 UCI datasets. In each panel, the six bars on the left correspond to an experiment with 'little' side-information $\mathcal{S}$, and the six on the right to 'much' side-information. From left to right, the six bars in each set are respectively K-means, K-means + diagonal metric, K-means + full metric, Constrained K-means (C-Kmeans), C-Kmeans + diagonal metric, and C-Kmeans + full metric. Also shown are $N$: size of dataset; $C$: number of classes/clusters; $d$: dimensionality of data; $K_c$: mean number of connected components (see footnotes 7, 9). 1 s.e. bars are also shown.

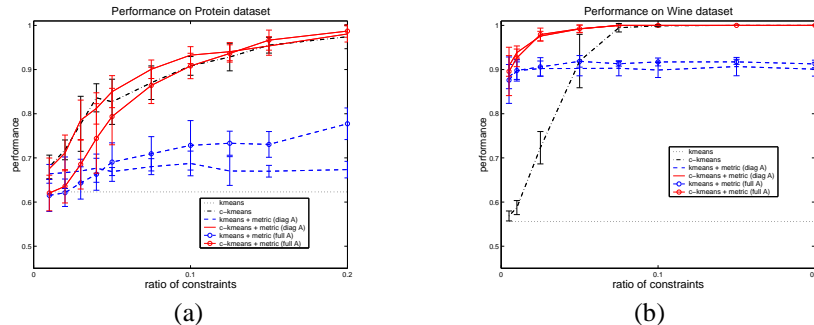

Figure 7: Plots of accuracy vs. amount of side-information. Here, the $x$-axis gives the fraction of all pairs of points in the same class that are randomly sampled to be included in $\mathcal{S}$.

margin. Not surprisingly, we also see that having more side-information in $\mathcal{S}$ typically leads to metrics giving better clusterings.

Figure 7 also shows two typical examples of how the quality of the clusterings found increases with the amount of side-information. For some problems (e.g., wine), our algorithm learns good diagonal and full metrics quickly with only a very small amount of side-information; for some others (e.g., protein), the distance metric, particularly the full metric, appears harder to learn and provides less benefit over constrained K-means.

## 4 Conclusions

We have presented an algorithm that, given examples of similar pairs of points in $\mathbb{R}^n$, learns a distance metric that respects these relationships. Our method is based on posing metric learning as a convex optimization problem, which allowed us to derive efficient, local-optima free algorithms. We also showed examples of diagonal and full metrics learned from simple artificial examples, and demonstrated on artificial and on UCI datasets how our methods can be used to improve clustering performance.

## Footnotes

[1]Technically, this also allows pseudometrics, where $d_A(x, y) = 0$ does not imply $x = y$.

[2]Note that, but putting the original dataset through a non-linear basis function $\phi$ and considering $\sqrt{(\phi(x) - \phi(y))^T A(\phi(x) - \phi(y))}$, non-linear distance metrics can also be learned.

[3]The proof is reminiscent of the derivation of Fisher's linear discriminant. Briefly, consider maximizing $\left( \sum_{(x_i, x_j) \in \mathcal{D}} ||x_i - x_j||_A^2 \right) / \sum_{(x_i, x_j) \in \mathcal{S}} ||x_i - x_j||_A^2 = \text{trace } A M_{\mathcal{D}} / \text{trace } A M_{\mathcal{S}}$, where $M_{\mathcal{T}} = \sum_{(x_i, x_j) \in \mathcal{T}} (x_i - x_j)(x_i - x_j)^T$. Decomposing $A$ as $A = \sum_{i=1}^n a_i a_i^T$ (always possible since $A \succeq 0$), this gives $\sum_i a_i^T M_{\mathcal{D}} a_i / \sum_i a_i^T M_{\mathcal{S}} a_i$, which we recognize as a Rayleigh-quotient like quantity whose solution is given by (say) solving the generalized eigenvector problem $M_{\mathcal{D}} a_1 = \lambda M_{\mathcal{S}} a_1$ for the principal eigenvector, and setting $a_2 = \ldots = a_n = 0$.

[4]To ensure that $A \succeq 0$, which is true iff the diagonal elements $A_{ii}$ are non-negative, we actually replace the Newton update $H^{-1} \nabla g$ by $\alpha H^{-1} \nabla g$, where $\alpha$ is a step-size parameter optimized via a line-search to give the largest downhill step subject to $A_{ii} \geq 0$.

[5]The algorithm shown in the figure includes a small refinement that the gradient step is taken the direction of the projection of $\nabla_A g$ onto the orthogonal subspace of $\nabla_A f$, so that it will "minimally" disrupt the constraint $C_1$. Empirically, this modification often significantly speeds up convergence.

[6]In the experiments with synthetic data, $\mathcal{S}$ was a randomly sampled 1% of all pairs of similar points.

[7]This is implemented as the usual K-means, except if $(x_i, x_j) \in \mathcal{S}$, then during the step in which points are assigned to cluster centroids $\mu_k$, we assign both $x_i$ and $x_j$ to cluster $\arg\min_k (x_i - \mu_k)^2 + (x_j - \mu_k)^2$. More generally, if we imagine drawing an edge between each pair of points in $S$, then all the points in each resulting connected component $C$ are constrained to lie in the same cluster, which we pick to be $\arg\min_k \sum_{x_i \in C} (x_i - \mu_k)^2$.

[8]In the case of many ($> 2$) clusters, this evaluation metric tends to give inflated scores since almost any clustering will correctly predict that most pairs are in different clusters. In this setting, we therefore modified the measure averaging not only $x_i$, $x_j$ drawn uniformly at random, but from the same cluster (as determined by $\hat{c}$) with chance 0.5, and from different clusters with chance 0.5, so that "matches" and "mis-matches" are given the same weight. All results reported here used K-means with multiple restarts, and are averages over at least 20 trials (except for wine, 10 trials).

[9]$\mathcal{S}$ was generated by picking a random subset of all pairs of points sharing the same class $c_i$. In the case of "little" side-information, the size of the subset was chosen so that the resulting number of resulting connected components $K_c$ (see footnote 7) would be very roughly 90% of the size of the original dataset. In the case of "much" side-information, this was changed to 70%.

## References

[1] C. Atkeson, A. Moore, and S. Schaal. Locally weighted learning. *AI Review*, 1996.

[2] T. Cox and M. Cox. *Multidimensional Scaling*. Chapman & Hall, London, 1994.

[3] C. Domeniconi and D. Gunopulos. Adaptive nearest neighbor classification using support vector machines. In *Advances in Neural Information Processing Systems 14*. MIT Press, 2002.

[4] G. H. Golub and C. F. Van Loan. *Matrix Computations*. Johns Hopkins Univ. Press, 1996.

[5] T. Hastie and R. Tibshirani. Discriminant adaptive nearest neighbor classification. *IEEE Transactions on Pattern Analysis and Machine Learning*, 18:607–616, 1996.

[6] T.S. Jaakkola and D. Haussler. Exploiting generative models in discriminaive classifier. In *Proc. of Tenth Conference on Advances in Neural Information Processing Systems*, 1999.

[7] I.T. Jolliffe. *Principal Component Analysis*. Springer-Verlag, New York, 1989.

[8] R. Rockafellar. *Convex Analysis*. Princeton Univ. Press, 1970.

[9] S.T. Roweis and L.K. Saul. Nonlinear dimensionality reduction by locally linear embedding. *Science 290*: 2323-2326.

[10] B. Scholkopf and A. Smola. *Learning with Kernels*. In Press, 2001.

[11] N. Tishby, F. Pereira, and W. Bialek. The information bottleneck method. In *Proc. of the 37th Allerton Conference on Communication, Control and Computing*, 1999.

[12] K. Wagstaff, C. Cardie, S. Rogers, and S. Schroedl. Constrained k-means clustering with background knowledge. In *Proc. 18th International Conference on Machine Learning*, 2001.
